# Action-Gap Phenomenon in Reinforcement Learning

**Amir-massoud Farahmand**[*]
School of Computer Science, McGill University
Montreal, Quebec, Canada

## Abstract

Many practitioners of reinforcement learning problems have observed that often-times the performance of the agent reaches very close to the optimal performance even though the estimated (action-)value function is still far from the optimal one. The goal of this paper is to explain and formalize this phenomenon by introducing the concept of the action-gap regularity. As a typical result, we prove that for an agent following the greedy policy $\hat{\pi}$ with respect to an action-value function $\hat{Q}$, the performance loss $\mathbb{E}\left[V^*(X) - V^{\hat{\pi}}(X)\right]$ is upper bounded by $O(\|\hat{Q} - Q^*\|_\infty^{1+\zeta})$, in which $\zeta \geq 0$ is the parameter quantifying the action-gap regularity. For $\zeta > 0$, our results indicate smaller performance loss compared to what previous analyses had suggested. Finally, we show how this regularity affects the performance of the family of approximate value iteration algorithms.

## 1   Introduction

This paper introduces a new type of regularity in the reinforcement learning (RL) and planning problems with finite-action spaces that suggests that the convergence rate of the performance loss to zero is faster than what previous analyses had indicated. The effect of this regularity, which we call the *action-gap regularity*, is that oftentimes the performance of the RL agent reaches very close to the optimal performance (e.g., it always solves the mountain-car problem with the optimal number of steps) even though the estimated action-value function is still far from the optimal one.

Figure 1 illustrates the effect of this regularity in a simple problem. We use value iteration to solve a stochastic 1D chain walk problem (slight modification of the example in Section 9.1 of [1]). The behavior of the supremum of the difference between the estimate after $k$ iterations and the optimal action-value function is $O(\gamma^k)$, in which $0 \leq \gamma < 1$ is the discount factor (notations shall be introduced in Section 2). The current theoretical results suggest that the convergence of the performance loss, which is defined as the average difference between the value of the optimal policy and the value of the greedy policy w.r.t. (with respect to) the estimated action-value function, should have the same $O(\gamma^k)$ behavior (cf. Proposition 6.1 of Bertsekas and Tsitsiklis [2]). However, the behavior of the performance loss is often considerably faster, e.g., it is approximately $O(\gamma^{1.85k})$ in this example.

To gain a better understanding of the action-gap regularity, focus on a single state and suppose that there are only two actions available. When the estimated action-value function has a large error, the greedy policy w.r.t. it can possibly choose the suboptimal action. However, when the error becomes smaller than the (half of the) gap between the value of the optimal action and the other one, the selected greedy action is the optimal action. After passing this threshold, the size of the error in the estimate of the action-value function in that state does not have any effect on the quality of the selected action. The larger the gap is, the more inaccurate the estimate can be while the selected greedy action is the optimal one. On the other hand, if the estimated action-value function does not suggest a correct ordering of actions but the gap is negligibly small, the performance loss of not

---

[*] www.SoloGen.net

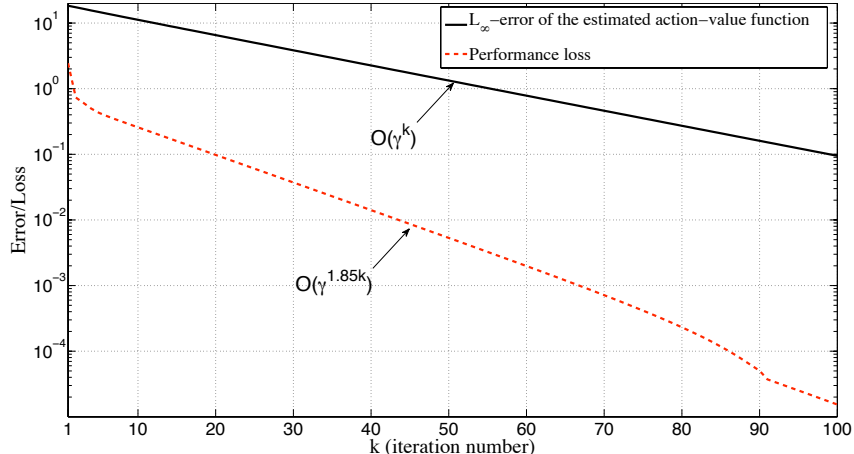

Figure 1: Comparison of the action-value estimation error $\|\hat{Q} - Q^*\|_\infty$ and the performance loss $\|V^* - V^{\hat{\pi}}\|_1$ ($\hat{\pi}$ is the greedy policy with respect to $\hat{Q}$) at different iterations of the value iteration algorithm. The rate of decrease for the performance loss is considerably faster than that of the estimation error. The problem is a 1D stochastic chain walk with 500 states and $\gamma = 0.95$.

choosing the optimal action is small as well. The presence of this gap in the optimal action-value function is what we call the action-gap regularity of the problem and the described behavior is called the *action-gap phenomenon*.

Action-gap regularity is similar to the *low-noise (or margin) condition* in the classification literature. The low-noise condition is the assumption that the conditional probability of the class label given input is "far" from the critical decision point. If this condition holds, "fast" convergence rate is obtainable as was shown by Mammen and Tsybakov [3], Tsybakov [4], Audibert and Tsybakov [5]. The low-noise condition is believed to be one reason that many high-dimensional classification problems can be solved with efficient sample complexity (cf. Rinaldo and Wasserman [6]). We borrow techniques developed in the classification literature, in particular by Audibert and Tsybakov [5], in our analysis.

It is notable that there have been some works that used classification algorithms to solve reinforcement learning (e.g., Lagoudakis and Parr [7], Lazaric et al. [8]) or the related problem of apprenticeship learning (e.g., Syed and Schapire [9]). Nevertheless, the connection of this work to the classification literature is only by borrowing theoretical ideas from that literature and not in using any particular algorithm. The focus of this work is indeed on the value-based approaches, though one might expect that similar behavior can be observed in classification-based approaches as well.

In the rest of this paper, we formalize the action-gap phenomenon and prove that whenever the MDP has a favorable action-gap regularity, fast convergence rate is achievable. Theorem 1 upper bounds the performance loss of the greedy policy w.r.t. the estimated action-value function by a function of the $L_p$-norm of the difference between the estimated action-value function and the optimal one. Our result complements previous theoretical analyses of RL/Planning problems such as those by Antos et al. [10], Munos and Szepesvári [11], Farahmand et al. [12, 13], Maillard et al. [14], who mainly focused on the quality of the (action-)value function estimate and ignored the action-gap regularity. This synergy provides a clearer picture of what makes an RL/Planning problem easy or difficult. Finally as an example of Theorem 1, we address the question of how the errors caused at each iteration of the Approximate Value Iteration (AVI) algorithm affect the quality of the outcome policy and show that the AVI procedure benefits from the action-gap regularity of the problem (Theorem 2).

## 2   Notations

In this section, we provide a brief summary of some of the concepts and definitions from the theory of MDPs and RL. For more information, the reader is referred to Bertsekas and Tsitsiklis [2], Sutton and Barto [15], Szepesvári [16].

For a space $\Omega$, with $\sigma$-algebra $\sigma_\Omega$, we define $\mathcal{M}(\Omega)$ as the set of all probability measures over $\sigma_\Omega$. $B(\Omega)$ denotes the space of bounded measurable functions w.r.t. (with respect to) $\sigma_\Omega$ and $B(\Omega, L)$ denotes the subset of $B(\Omega)$ with bound $0 < L < \infty$.

A *finite-action discounted MDP* is a 5-tuple $(\mathcal{X}, \mathcal{A}, P, \mathcal{R}, \gamma)$, where $\mathcal{X}$ is a measurable state space, $\mathcal{A}$ is a finite set of actions, $P : \mathcal{X} \times \mathcal{A} \to \mathcal{M}(\mathcal{X})$ is the transition probability kernel, $\mathcal{R} : \mathcal{X} \times \mathcal{A} \to \mathbb{R}$ is the reward distribution, and $0 \leq \gamma < 1$ is a discount factor. We denote $r(x,a) = \mathbb{E}\left[\mathcal{R}(\cdot|x,a)\right]$.

A measurable mapping $\pi : \mathcal{X} \to \mathcal{A}$ is called a deterministic Markov stationary policy, or just a *policy* in short. An agent's following a policy $\pi$ in an MDP means that at each time step $A_t = \pi(X_t)$.

A policy $\pi$ induces two transition probability kernels $P^\pi : \mathcal{X} \to \mathcal{M}(\mathcal{X})$ and $P^\pi : \mathcal{X} \times \mathcal{A} \to \mathcal{M}(\mathcal{X} \times \mathcal{A})$. For a measurable subset $A$ of $\mathcal{X}$ and a measurable subset $B$ of $\mathcal{X} \times \mathcal{A}$, we define $(P^\pi)(A|x) \triangleq \int P(dy|x, \pi(x)) \mathbb{I}_{\{y \in A\}}$ and $(P^\pi)(B|x,a) \triangleq \int P(dy|x,a) \mathbb{I}_{\{(y, \pi(y)) \in B\}}$. The $m$-step transition probability kernel $(P^\pi)^m : \mathcal{X} \times \mathcal{A} \to \mathcal{M}(\mathcal{X} \times \mathcal{A})$ for $m = 2, 3, \cdots$ are inductively defined as $(P^\pi)^m(B|x,a) \triangleq \int_\mathcal{X} P(dy|x,a)(P^\pi)^{m-1}(B|y, \pi(y))$ (similarly for $(P^\pi)^m : \mathcal{X} \to \mathcal{M}(\mathcal{X})$).

Given a transition probability kernel $P : \mathcal{X} \to \mathcal{M}(\mathcal{X})$, define the right-linear operator $P\cdot : B(\mathcal{X}) \to B(\mathcal{X})$ by $(PV)(x) \triangleq \int_\mathcal{X} P(dy|x)V(y)$. For a probability measure $\rho \in \mathcal{M}(\mathcal{X})$ and a measurable subset $A$ of $\mathcal{X}$, define the left-linear operators $\cdot P : \mathcal{M}(\mathcal{X}) \to \mathcal{M}(\mathcal{X})$ by $(\rho P)(A) = \int \rho(dx) P(dy|x) \mathbb{I}_{\{y \in A\}}$. A typical choice of $P$ is $(P^\pi)^m : \mathcal{M}(\mathcal{X}) \to \mathcal{M}(\mathcal{X})$. These operators for $P : \mathcal{X} \times \mathcal{A} \to \mathcal{M}(\mathcal{X} \times \mathcal{A})$ are defined similarly.

The value function $V^\pi$ and and the action-value function $Q^\pi$ of a policy $\pi$ are defined as follows: Let $(R_t; t \geq 1)$ be the sequence of rewards when the Markov chain is started from state $X_1$ (state-action $(X_1, A_1)$ for the action-value function) drawn from a positive probability distribution over $\mathcal{X}$ ($\mathcal{X} \times \mathcal{A}$) and the agent follows the policy $\pi$. Then $V^\pi(x) \triangleq \mathbb{E}\left[\sum_{t=1}^\infty \gamma^{t-1} R_t \,\middle|\, X_1 = x\right]$ and $Q^\pi(x,a) \triangleq \mathbb{E}\left[\sum_{t=1}^\infty \gamma^{t-1} R_t \,\middle|\, X_1 = x, A_1 = a\right]$.

For a discounted MDP, we define the *optimal value* and *optimal action-value* functions by $V^*(x) = \sup_\pi V^\pi(x)$ for all states $x \in \mathcal{X}$ and $Q^*(x,a) = \sup_\pi Q^\pi(x,a)$ for all state-actions $(x,a) \in \mathcal{X} \times \mathcal{A}$. We say that a policy $\pi^*$ is *optimal* if it achieves the best values in every state, i.e., if $V^{\pi^*} = V^*$. We say that a policy $\pi$ is *greedy* w.r.t. an action-value function $Q$ and write $\pi = \hat{\pi}(\cdot; Q)$, if $\pi(x) = \text{argmax}_{a \in \mathcal{A}} Q(x,a)$ holds for all $x \in \mathcal{X}$ (if there exist multiple maximizers, a maximizer is chosen in an arbitrary deterministic manner). Greedy policies are important because a greedy policy w.r.t. the optimal action-value function $Q^*$ is an optimal policy.

For a fixed policy $\pi$, the Bellman operators $T^\pi : B(\mathcal{X}) \to B(\mathcal{X})$ and $T^\pi : B(\mathcal{X} \times \mathcal{A}) \to B(\mathcal{X} \times \mathcal{A})$ (for the action-value functions) are defined as $(T^\pi V)(x) \triangleq r(x, \pi(x)) + \gamma \int_\mathcal{X} V(y) P(dy|x, \pi(x))$ and $(T^\pi Q)(x,a) \triangleq r(x,a) + \gamma \int_\mathcal{X} Q(y, \pi(y)) P(dy|x,a)$. The fixed point of the Bellman operator is the (action-)value function of the policy $\pi$, i.e., $T^\pi Q^\pi = Q^\pi$ and $T^\pi V^\pi = V^\pi$. Similarly, the Bellman optimality operators $T^* : B(\mathcal{X}) \to B(\mathcal{X})$ and $T^* : B(\mathcal{X} \times \mathcal{A}) \to B(\mathcal{X} \times \mathcal{A})$ (for the action-value functions) are defined as $(T^* V)(x) \triangleq \max_a \left\{ r(x,a) + \gamma \int_{\mathbb{R} \times \mathcal{X}} V(y) P(dr, dy|x,a) \right\}$ and $(T^* Q)(x,a) \triangleq r(x,a) + \gamma \int_{\mathbb{R} \times \mathcal{X}} \max_{a'} Q(y, a') P(dr, dy|x,a)$. Again, these operators enjoy a fixed-point property similar to that of the Bellman operators: $T^* Q^* = Q^*$ and $T^* V^* = V^*$.

For a probability measure $\rho \in \mathcal{M}(\mathcal{X})$, and a measurable function $V \in B(\mathcal{X})$, we define the $L_p(\rho)$-norm ($1 \leq p < \infty$) of $V$ as $\|V\|_{p,\rho} \triangleq \left[\int_\mathcal{X} |V(x)|^p \, d\rho(x)\right]^{1/p}$. The $L_\infty(\mathcal{X})$-norm is defined as $\|V\|_\infty \triangleq \sup_{x \in \mathcal{X}} |V(x)|$. For $\rho \in \mathcal{M}(\mathcal{X} \times \mathcal{A})$ and $Q \in B(\mathcal{X} \times \mathcal{A})$, we define $\|Q\|_{p,\rho}$ ($1 \leq p < \infty$) by $\|Q\|_{p,\rho} \triangleq \left[\frac{1}{|\mathcal{A}|} \sum_{a=1}^{|\mathcal{A}|} \|Q(\cdot, a)\|_{p,\rho}^p\right]^{1/p}$ and $\|Q\|_\infty \triangleq \sup_{(x,a) \in \mathcal{X} \times \mathcal{A}} |Q(x,a)|$.

## 3  Action-Gap Theorem

In this section, we present the action-gap theorem for an MDP $(\mathcal{X}, \mathcal{A}, P, \mathcal{R}, \gamma)$. To simplify the analysis, we assume that the number of actions $|\mathcal{A}|$ is only 2. We denote $\rho^* \in \mathcal{M}(\mathcal{X})$ as the station-

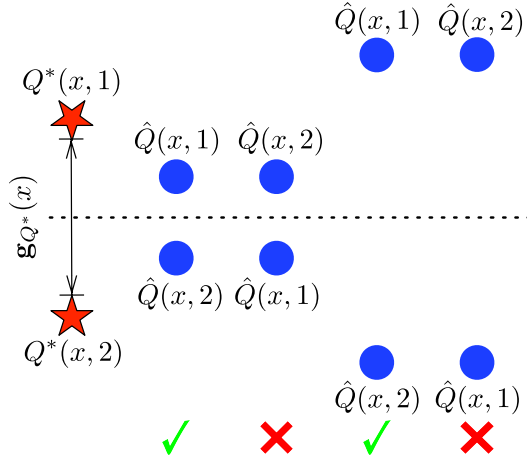

Figure 2: The action-gap function $\mathbf{g}_{Q^*}(x)$ and the relative ordering of the optimal and the estimated action-value functions for a single state $x$. Depending on the ordering of the estimates, the greedy action is the same as (✓) or different from (✗) the optimal action. This figure does not show all possible configurations.

ary distribution induced by $\pi^*$, and we let $\rho \in \mathcal{M}(\mathcal{X})$ be a user-specified *evaluation* distribution. This distribution indicates the relative importance of regions of the state space to the user.

Suppose that algorithm $\mathsf{A}$ receives a dataset $\mathcal{D}_n = \{(X_1, A_1, R_1, X_1'), \dots, (X_n, A_n, R_n, X_n')\}$ (with $R_i$ is being drawn from $\mathcal{R}(\cdot | X_t, A_t)$ and $X_t'$ is being drawn from $P(\cdot | X_t, A_t)$) and outputs $\hat{Q}$ as an estimate of the optimal action-value function, i.e., $\hat{Q} \leftarrow \mathsf{A}(\mathcal{D}_n)$. The exact nature of this algorithm is not important and it can be any online or offline, batch or incremental algorithms of choice such as Q-learning, SARSA [15], and their variants [17], LSPI [1], LARS-TD [18] in a policy iteration procedure, REG-LSPI [13], various Fitted Q-Iterations algorithms [19, 20, 12], or Linear Programming-based approaches [21, 22]. The only relevant aspect of $\hat{Q}$ is how well it approximates $Q^*$. We quantify the quality of the approximation by the $L_p$-norm $\|\hat{Q} - Q^*\|_{p,\rho^*}$ ($p \in [1, \infty]$).

The *performance loss* (or *regret*) of a policy $\pi$ is the expected difference between the value of the optimal policy $\pi^*$ to the value of $\pi$ when the initial state distribution is selected according to $\rho$, i.e.,

$$\mathrm{Loss}(\pi; \rho) \triangleq \int_{\mathcal{X}} (V^*(x) - V^\pi(x)) \, \mathrm{d}\rho(x). \tag{1}$$

The value of $\mathrm{Loss}(\hat{\pi}; \rho)$, in which $\hat{\pi}$ is the greedy policy w.r.t. $\hat{Q}$, is the main quantity of interest and indicates how much worse the agent following policy $\hat{\pi}$ would perform, in average, compared to the optimal one. The choice of $\rho$ enables the user to specify the relative importance of regions in the state space.

We define the *action(-value)-gap function* $\mathbf{g}_{Q^*} : \mathcal{X} \to \mathbb{R}$ as

$$\mathbf{g}_{Q^*}(x) \triangleq |Q^*(x, 1) - Q^*(x, 2)|.$$

This gap is shown in Figure 2. The following assumption quantifies the action-gap regularity.

**Assumption A1 (Action-Gap).** For a fixed MDP $(\mathcal{X}, \mathcal{A}, P, \mathcal{R}, \gamma)$ with $|\mathcal{A}| = 2$, there exist constants $c_g > 0$ and $\zeta \geq 0$ such that for all $t > 0$, we have

$$\mathbb{P}_{\rho^*}(0 < \mathbf{g}_{Q^*}(X) \leq t) \triangleq \int_{\mathcal{X}} \mathbb{I}\{0 < \mathbf{g}_{Q^*}(x) \leq t\} \, \mathrm{d}\rho^*(x) \leq c_g \, t^\zeta.$$

The value of $\zeta$ controls the distribution of the action-gap $\mathbf{g}_{Q^*}(X)$. A large value of $\zeta$ indicates that the probability that $Q(X, 1)$ being very close to $Q(X, 2)$ is small and vice versa. The smallness of this probability implies that the estimated action-value function $\hat{Q}$ might be rather inaccurate in a

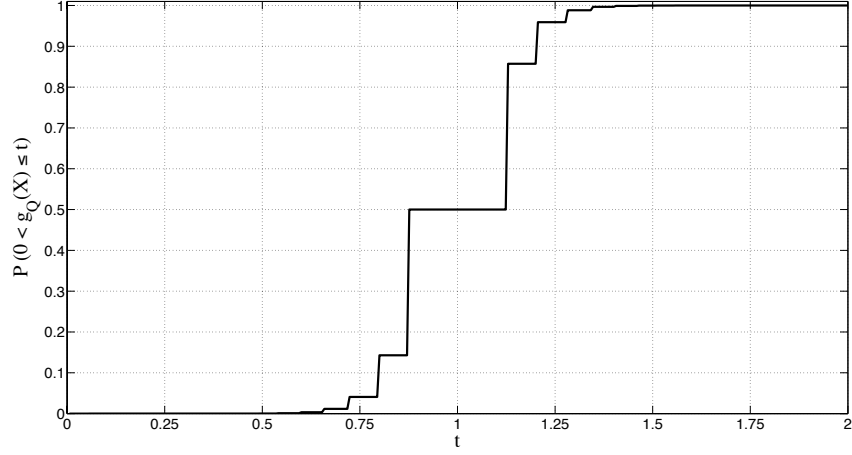

Figure 3: The probability distribution $\mathbb{P}_{\rho^*}\left(0 < \mathbf{g}_{Q^*}(X) \leq t\right)$ for a 1D stochastic chain walk with $500$ states and $\gamma = 0.95$. Here the probability of the action-gap being close to zero is small.

large subset of the state space (measured according to $\rho^*$) but its corresponding greedy policy would still be the same as the optimal one. The case of $\zeta = 0$ and $c_g = 1$ is equivalent to not having any assumption on the action-gap. This assumption is inspired by the low-noise condition in the classification literature [5]. As an example of a typical behavior of an action-gap function, Figure 3 depicts $\mathbb{P}_{\rho^*}\left(0 < \mathbf{g}_{Q^*}(X) \leq t\right)$ for the same 1D stochastic chain walk problem as mentioned in the Introduction. It is seen that the probability that the action-gap function $\mathbf{g}_{Q^*}$ being close to zero is very small. Note that the specific polynomial form of the upper bound in Assumption A1 is only a modeling assumption that captures the essence of the action-gap regularity without trying to be too general to lead to unnecessarily complicated analyses.

As a result of the dynamical nature of MDP, the performance loss depends not only on the choice of $\rho$ and $\rho^*$, but also on the transition probability kernel $P$. To analyze this dependence, we define a *concentrability* coefficient and use a change of measure argument similar to the work of Munos [23, 24], Antos et al. [10].

**Definition 1** (Concentrability of the Future-State Distribution). *Given $\rho, \rho^* \in \mathcal{M}(\mathcal{X})$, a policy $\pi$, and an integer $m \geq 0$, let $\rho(P^\pi)^m \in \mathcal{M}(\mathcal{X})$ denote the future-state distribution obtained when the first state is distributed according to $\rho$ and we then follow the policy $\pi$ for $m$ steps. Denote the supremum of the Radon-Nikodym derivative of $\rho(P^\pi)^m$ w.r.t. $\rho^*$ by $c(m; \pi)$, i.e.,*

$$c(m; \pi) \triangleq \left\| \frac{\mathrm{d}(\rho(P^\pi)^m)}{\mathrm{d}\rho^*} \right\|_\infty.$$

*If $\rho(P^\pi)^m$ is not absolutely continuous w.r.t. $\rho^*$, we set $c(m; \pi) = \infty$. The concentrability of the future-state distribution coefficient is defined as*

$$C(\rho, \rho^*) \triangleq \sup_\pi \sum_{m \geq 0} \gamma^m c(m; \pi).$$

The following theorem upper bounds the performance loss $\mathrm{Loss}(\hat{\pi}; \rho)$ as a function of $\|Q^* - \hat{Q}\|_{p, \rho^*}$, the action-gap distribution, and the concentrability coefficient.

**Theorem 1.** *Consider an MDP $(\mathcal{X}, \mathcal{A}, P, \mathcal{R}, \gamma)$ with $|\mathcal{A}| = 2$ and an estimate $\hat{Q}$ of the optimal action-value function. Let Assumption A1 hold and $C(\rho, \rho^*) < \infty$. Denote $\hat{\pi}$ as the greedy policy w.r.t. $\hat{Q}$. We then have*

$$\mathrm{Loss}(\hat{\pi}; \rho) \leq \begin{cases} 2^{1+\zeta}\, c_g\, C(\rho, \rho^*) \left\| \hat{Q} - Q^* \right\|_\infty^{1+\zeta}, \\ 2^{1+\frac{p(1+\zeta)}{p+\zeta}}\, c_g^{\frac{p-1}{p+\zeta}}\, C(\rho, \rho^*) \left\| \hat{Q} - Q^* \right\|_{p,\rho^*}^{\frac{p(1+\zeta)}{p+\zeta}}. \qquad (1 \leq p < \infty) \end{cases}$$

*Proof.* Let function $F : \mathcal{X} \to \mathbb{R}$ be defined as $F(x) = V^*(x) - V^{\hat{\pi}}(x) = Q^{\pi^*}(x, \pi^*(x)) - Q^{\hat{\pi}}(x, \hat{\pi}(x))$ for any $x \in \mathcal{X}$. Note that $\mathrm{Loss}(\hat{\pi}; \rho) = \rho F$. Decompose $F(x)$ as

$$F(x) = \left( Q^{\pi^*}(x, \pi^*(x)) - Q^{\pi^*}(x, \hat{\pi}(x)) \right) + \left( Q^{\pi^*}(x, \hat{\pi}(x)) - Q^{\hat{\pi}}(x, \hat{\pi}(x)) \right) = F_1(x) + F_2(x).$$

We have

$$F_2(x) = \left[ r(x, \hat{\pi}(x)) + \gamma \int_{\mathcal{X}} P(dy|x, \hat{\pi}(x)) Q^{\pi^*}(y, \pi^*(y)) \right] -$$
$$\left[ r(x, \hat{\pi}(x)) + \gamma \int_{\mathcal{X}} P(dy|x, \hat{\pi}(x)) Q^{\hat{\pi}}(y, \hat{\pi}(y)) \right]$$
$$= \gamma P^{\hat{\pi}}(\cdot|x) F(\cdot).$$

Therefore, $F = (\mathbf{I} - \gamma P^{\hat{\pi}})^{-1} F_1 = \sum_{m \geq 0} (\gamma P^{\hat{\pi}})^m F_1$. Thus,

$$\rho F = \sum_{m \geq 0} \rho (\gamma P^{\hat{\pi}})^m F_1 = \sum_{m \geq 0} \gamma^m \int_{\mathcal{X}} \left( \rho (P^{\hat{\pi}})^m \right) (dy) F_1(y)$$
$$= \sum_{m \geq 0} \gamma^m \int_{\mathcal{X}} \frac{d \left( \rho (P^{\hat{\pi}})^m \right)}{d\rho^*}(y) d\rho^*(y) F_1(y)$$
$$\leq \sum_{m \geq 0} \gamma^m c(m; \hat{\pi}) \rho^* F_1 \leq C(\rho, \rho^*) \rho^* F_1. \tag{2}$$

in which we used the Radon-Nikodym theorem and the definition of concentrability coefficient. Let us turn to $F_1$ and provide an upper bound for it. We use techniques similar to [5].

$L_\infty$ **result:** Note that for any given $x \in \mathcal{X}$, if for some value of $\varepsilon > 0$ we have $\hat{\pi}(x) \neq \pi^*(x)$ and $|Q^{\pi^*}(x, a) - \hat{Q}(x, a)| \leq \varepsilon$ (for both $a = 1, 2$), then it holds that $\mathbf{g}_{Q^*}(x) = |Q^{\pi^*}(x, 1) - Q^{\pi^*}(x, 2)| \leq 2\varepsilon$. To show it, suppose that instead $\mathbf{g}_{Q^*}(x) = |Q^{\pi^*}(x, 1) - Q^{\pi^*}(x, 2)| > 2\varepsilon$. Then because of the assumption $|Q^{\pi^*}(x, a) - \hat{Q}(x, a)| \leq \varepsilon$ ($a = 1, 2$), the ordering of $\hat{Q}(x, 1)$ and $\hat{Q}(x, 2)$ is the same as the ordering of $Q^*(x, 1)$ and $Q^*(x, 2)$, which contradicts the assumption that $\hat{\pi}(x) \neq \pi^*(x)$ (see Figure 2).

Denote $\varepsilon_0 = \|Q^{\pi^*} - \hat{Q}\|_\infty$. Whenever $\hat{\pi}(x) = \pi^*(x)$, the value of $F_1(x)$ is zero, so we get

$$F_1(x) = \left[ Q^{\pi^*}(x, \pi^*(x)) - Q^{\pi^*}(x, \hat{\pi}(x)) \right] \left[ \mathbb{I}\{\hat{\pi}(x) = \pi^*(x)\} + \mathbb{I}\{\hat{\pi}(x) \neq \pi^*(x)\} \right]$$
$$= \left[ Q^{\pi^*}(x, \pi^*(x)) - Q^{\pi^*}(x, 1 - \pi^*(x)) \right] \mathbb{I}\{\hat{\pi}(x) \neq \pi^*(x)\}$$
$$\times \left[ \mathbb{I}\{\mathbf{g}_{Q^*}(x) = 0\} + \mathbb{I}\{0 < \mathbf{g}_{Q^*}(x) \leq 2\varepsilon_0\} + \mathbb{I}\{\mathbf{g}_{Q^*}(x) > 2\varepsilon_0\} \right]$$
$$\leq 0 + 2\varepsilon_0 \mathbb{I}\{0 < \mathbf{g}_{Q^*}(x) \leq 2\varepsilon_0\} + 0.$$

Here we used the definition of $\mathbf{g}_{Q^*}(x)$ and the fact that $\mathbf{g}_{Q^*}(x)$ is no larger than $2\varepsilon_0$. This result together with Assumption A1 show that $\rho^* F_1 \leq 2\varepsilon_0 \mathbb{P}_{\rho^*} (0 < \mathbf{g}_{Q^*}(X) \leq 2\varepsilon_0) \leq 2\varepsilon_0 c_g (2\varepsilon_0)^\varsigma$. Plugging this result in (2) finishes the proof of the first part.

$L_p$ **result:** For any given $x \in \mathcal{X}$, let $D(x) = |Q^{\pi^*}(x, 1) - \hat{Q}(x, 1)| + |Q^{\pi^*}(x, 2) - \hat{Q}(x, 2)|$. Whenever $\hat{\pi}(x) \neq \pi^*(x)$, we have $\mathbf{g}_{Q^*}(x) \leq D(x)$. Similar to the previous case, we have

$$F_1(x) = \left[ Q^{\pi^*}(x, \pi^*(x)) - Q^{\pi^*}(x, 1 - \pi^*(x)) \right] \mathbb{I}\{\hat{\pi}(x) \neq \pi^*(x)\}$$
$$\times \left[ \mathbb{I}\{\mathbf{g}_{Q^*}(x) = 0\} + \mathbb{I}\{0 < \mathbf{g}_{Q^*}(x) \leq t\} + \mathbb{I}\{\mathbf{g}_{Q^*}(x) > t\} \right]$$
$$\leq D(x) \left[ \mathbb{I}\{0 < \mathbf{g}_{Q^*}(x) \leq t\} + \mathbb{I}\{\mathbf{g}_{Q^*}(x) > t\} \right]$$

Take expectation w.r.t. $\rho^*$ and use Hölder's inequality to get

$$\rho^* F_1 \leq \|D\|_{p, \rho^*} \left[ \mathbb{P}_{\rho^*} (0 < \mathbf{g}_{Q^*}(X) \leq t) \right]^{\frac{p-1}{p}} + \|D\|_{p, \rho^*} \left[ \mathbb{P}_{\rho^*} (\mathbf{g}_{Q^*}(X) > t) \right]^{\frac{p-1}{p}}$$
$$\leq \|D\|_{p, \rho^*} \left( c_g t^\varsigma \right)^{\frac{p-1}{p}} + \|D\|_{p, \rho^*} \left[ \mathbb{P}_{\rho^*} (D(X) > t) \right]^{\frac{p-1}{p}}$$
$$\leq \|D\|_{p, \rho^*} \left( c_g t^\varsigma \right)^{\frac{p-1}{p}} + \frac{\|D\|_{p, \rho^*}^p}{t^{p-1}}.$$

where we used Assumption A1 and the definition of $D(\cdot)$ in the second inequality, and Markov's inequality in the last one. Minimize the upper bound in $t$ to get $t = c_g^{\frac{-1}{p+\zeta}} \|D\|_{p,\rho^*}^{\frac{p}{p+\zeta}}$. This leads to $\rho^* F_1 \leq 2c_g^{\frac{p-1}{p+\zeta}} \|D\|_{p,\rho^*}^{\frac{p(1+\zeta)}{p+\zeta}}$, which in turn alongside inequality (2) and $\|D\|_{p,\rho^*}^p \leq 2^p \|Q^{\pi^*} - \hat{Q}\|_{p,\rho^*}^p$ proves the second part of this result. $\qquad\square$

This theorem indicates that if $\|\hat{Q} - Q^*\|_p$ ($1 < p \leq \infty$) has an error upper bound of $O(n^{-\beta})$ (with $\beta$ typically in the range of $(0, 1/2]$ depending on the properties of the MDP and the estimator), we obtain faster convergence upper bounds on the performance loss $\text{Loss}(\hat{\pi}; \rho)$ whenever the problem has an action-gap regularity ($\zeta > 0$).

One might compare Theorem 1 with classical upper bounds such as $\|V^{\hat{\pi}} - V^{\pi^*}\|_\infty \leq \frac{2\gamma}{1-\gamma}\|\hat{V} - V^*\|_\infty$ (Proposition 6.1 of Bertsekas and Tsitsiklis [2]). In order to make these two bounds comparable, we slightly modify the proof of our theorem to get the $L_\infty$-norm in the left hand side. The result would be $\|V^* - V^{\hat{\pi}}\|_\infty \leq \frac{2^{1+\zeta}c_g}{1-\gamma}\|\hat{Q} - Q^*\|_\infty^{1+\zeta}$. If there is no action-gap assumption ($\zeta = 0$ and $c_g = 1$), the results are similar (except for a factor of $\gamma$ and that we measure the error by the difference in the action-value function as opposed to the value function), but when $\zeta > 0$ the error bound significantly improves.

## 4 Application of the Action-Gap Theorem in Approximate Value Iteration

The goal of this section is to show how the analysis based on the action-gap phenomenon might lead to a tighter upper bound on the performance loss for the family of the AVI algorithms. There are various AVI algorithms (Riedmiller [19], Ernst et al. [20], Munos and Szepesvári [11], Farahmand et al. [12]), that work by generating a sequence of action-value function estimates $(\hat{Q}_k)_{k=0}^K$, in which $\hat{Q}_{k+1}$ is the result of approximately applying the Bellman optimality operator to the previous estimate $\hat{Q}_k$, i.e., $\hat{Q}_{k+1} \approx T^*\hat{Q}_k$. Let us denote the error caused at each iteration by

$$\varepsilon_k \triangleq T^*\hat{Q}_k - \hat{Q}_{k+1}. \tag{3}$$

The following theorem, which is based on Theorem 3 of Farahmand et al. [25], relates the performance loss $\|Q^{\hat{\pi}(\cdot;\hat{Q}_K)} - Q^*\|_{1,\rho}$ of the obtained greedy policy $\hat{\pi}(\cdot; \hat{Q}_K)$ to the error sequence $(\varepsilon_k)_{k=0}^{K-1}$ and the action-gap assumption on the MDP. Before stating the theorem, we define the following sequence:

$$\alpha_k = \begin{cases} \frac{(1-\gamma)}{1-\gamma^{K+1}}\gamma^{K-k-1} & 0 \leq k < K, \\ \frac{(1-\gamma)}{1-\gamma^{K+1}}\gamma^K & k = K. \end{cases}$$

This sequence has $\alpha_k \propto \gamma^{K-k}$ behavior and satisfies $\sum_{k=0}^K \alpha_k = 1$.

**Theorem 2** (Error Propagation for AVI). *Consider an MDP $(\mathcal{X}, \mathcal{A}, P, \mathcal{R}, \gamma)$ with $|\mathcal{A}| = 2$ that satisfies Assumption A1 and has $C(\rho, \rho^*) < \infty$. Let $p \geq 1$ be a real number and $K$ be a positive integer. Then for any sequence $(\hat{Q}_k)_{k=0}^K \subset B(\mathcal{X} \times \mathcal{A}, Q_{max})$ and the corresponding sequence $(\varepsilon_k)_{k=0}^{K-1}$ defined in (3), we have*

$$\text{Loss}(\hat{\pi}(\cdot, Q_K); \rho) \leq 2\left(\frac{2}{1-\gamma}\right)^{\frac{p(1+\zeta)}{p+\zeta}} c_g^{\frac{p-1}{p+\zeta}} C(\rho, \rho^*) \left[\sum_{k=0}^{K-1} \alpha_k \|\varepsilon_k\|_{p,\rho^*}^p + \alpha_K(2Q_{max})^p\right]^{\frac{1+\zeta}{p+\zeta}}.$$

*Proof.* Similar to Lemma 4.1 of Munos [24], one may derive

$$Q^* - \hat{Q}_{k+1} = T^{\pi^*}Q^* - T^{\pi^*}\hat{Q}_k + T^{\pi^*}\hat{Q}_k - T^*\hat{Q}_k + \varepsilon_k \leq \gamma P^{\pi^*}(Q^* - \hat{Q}_k) + \varepsilon_k$$

where we used the property of the Bellman optimality operator $T^*\hat{Q}_k \geq T^{\pi^*}\hat{Q}_k$ and the definition of $\varepsilon_k$ (3). By induction, we get

$$Q^* - \hat{Q}_K \leq \sum_{k=0}^{K-1} \gamma^{K-k-1}(P^{\pi^*})^{K-k-1}\varepsilon_k + \gamma^K(P^{\pi^*})^K(Q^* - \hat{Q}_0).$$

Therefore, for any $p \geq 1$, the value of $\|Q^* - \hat{Q}_K\|_{p,\rho^*} = \rho^*|Q^* - \hat{Q}_K|^p$ is upper bounded by

$$\rho^*|Q^* - \hat{Q}_K|^p \leq \left(\frac{1-\gamma^{K+1}}{1-\gamma}\right)^p \left[\sum_{k=0}^{K-1} \alpha_k \rho^*(P^{\pi^*})^{K-k-1}|\varepsilon_k| + \alpha_K \rho^*(P^{\pi^*})^K|Q^* - \hat{Q}_0|\right]^p$$

$$\leq \left(\frac{1-\gamma^{K+1}}{1-\gamma}\right)^p \left[\sum_{k=0}^{K-1} \alpha_k \|\varepsilon_k\|_{p,\rho^*}^p + \alpha_K(2Q_{\max})^p\right],$$

where we used $\rho^*(P^{\pi^*})^m = \rho^*$ (for any $m \geq 0$) and Jensen's inequality. The application of Theorem 1 and noting that $(1-\gamma^{K+1})/(1-\gamma) \leq 1/(1-\gamma)$ lead to the desired result. $\qquad\square$

Comparing this theorem with Theorem 3 of Farahmand et al. [25] is instructive. Denoting $\mathcal{E} = \sum_{k=0}^{K-1} \alpha_k \|\varepsilon_k\|_{2,\rho^*}^2$, this paper's result indicates that the effect of the size of $\varepsilon_k$ on $\mathrm{Loss}(\hat{\pi}(\cdot,\hat{Q}_K);\rho)$ depends on $\mathcal{E}^{\frac{1+\zeta}{2+\zeta}}$, while [25], which does not consider the action-gap regularity, suggests that the effect depends on $\mathcal{E}^{1/2}$. For $\zeta > 0$, this indicates a faster convergence rate for the performance loss while for $\zeta = 0$, they are the same.

## 5   Conclusion

This work introduced the action-gap regularity in reinforcement learning and planning problems and analyzed the action-gap phenomenon for two-action discounted MDPs. We showed that when the problem has a favorable action-gap regularity, quantified by the parameter $\zeta$, the performance loss is much smaller than the error of the estimated optimal action-value function. The action-gap regularity, among other regularities such as the smoothness of the action-value function [13], is a step forward to better understanding of what properties of a sequential decision-making problem makes learning and planning easy or difficult.

There are several issues that deserve to be studied in the future. Among them is the extension of the current framework to multi-action discounted MDPs. Also it is important to study the relation between the parameter $\zeta$ of the action-gap regularity assumption to the properties of the MDP such as the transition probability kernel and the reward distribution.

## Acknowledgments

I thank the anonymous reviewers for their useful comments. This work was partly supported by AICML and NSERC.

## References

[1] Michail G. Lagoudakis and Ronald Parr. Least-squares policy iteration. *Journal of Machine Learning Research*, 4:1107–1149, 2003.

[2] Dimitri P. Bertsekas and John N. Tsitsiklis. *Neuro-Dynamic Programming (Optimization and Neural Computation Series, 3)*. Athena Scientific, 1996.

[3] Enno Mammen and Alexander B. Tsybakov. Smooth discrimination analysis. *The Annals of Statistics*, 27(6):1808–1829, 1999.

[4] Alexander B. Tsybakov. Optimal aggregation of classifiers in statistical learning. *The Annals of Statistics*, 32 (1):135–166, 2004.

[5] Jean-Yves Audibert and Alexander B. Tsybakov. Fast learning rates for plug-in classifiers. *The Annals of Statistics*, 35(2):608–633, 2007.

[6] Alessandro Rinaldo and Larry Wasserman. Generalized density clustering. *The Annals of Statistics*, 38(5):2678–2722, 2010.

[7] Michail G. Lagoudakis and Ronald Parr. Reinforcement learning as classification: Leveraging modern classifiers. In *ICML '03: Proceedings of the 20th international conference on Machine learning*, pages 424–431, 2003.

[8] Alessandro Lazaric, Mohammad Ghavamzadeh, and Rémi Munos. Analysis of a classification-based policy iteration algorithm. In *Proceedings of the 27th International Conference on Machine Learning (ICML-10)*, pages 607–614. Omnipress, 2010.

[9] Omar Syed and Robert E. Schapire. A reduction from apprenticeship learning to classification. In J. Lafferty, C. K. I. Williams, J. Shawe-Taylor, R.S. Zemel, and A. Culotta, editors, *Advances in Neural Information Processing Systems (NIPS - 23)*, pages 2253–2261, 2010.

[10] András Antos, Csaba Szepesvári, and Rémi Munos. Learning near-optimal policies with Bellman-residual minimization based fitted policy iteration and a single sample path. *Machine Learning*, 71:89–129, 2008.

[11] Rémi Munos and Csaba Szepesvári. Finite-time bounds for fitted value iteration. *Journal of Machine Learning Research*, 9:815–857, 2008.

[12] Amir-massoud Farahmand, Mohammad Ghavamzadeh, Csaba Szepesvári, and Shie Mannor. Regularized fitted Q-iteration for planning in continuous-space Markovian Decision Problems. In *Proceedings of American Control Conference (ACC)*, pages 725–730, June 2009.

[13] Amir-massoud Farahmand, Mohammad Ghavamzadeh, Csaba Szepesvári, and Shie Mannor. Regularized policy iteration. In D. Koller, D. Schuurmans, Y. Bengio, and L. Bottou, editors, *Advances in Neural Information Processing Systems (NIPS - 21)*, pages 441–448. MIT Press, 2009.

[14] Odalric Maillard, Rémi Munos, Alessandro Lazaric, and Mohammad Ghavamzadeh. Finite-sample analysis of Bellman residual minimization. In *Proceedings of the Second Asian Conference on Machine Learning (ACML)*, 2010.

[15] Richard S. Sutton and Andrew G. Barto. *Reinforcement Learning: An Introduction (Adaptive Computation and Machine Learning)*. The MIT Press, 1998.

[16] Csaba Szepesvári. *Algorithms for Reinforcement Learning*. Morgan Claypool Publishers, 2010.

[17] Hamid Reza Maei, Csaba Szepesvári, Shalabh Bhatnagar, and Richard S. Sutton. Toward off-policy learning control with function approximation. In Johannes Fürnkranz and Thorsten Joachims, editors, *Proceedings of the 27th International Conference on Machine Learning (ICML-10)*, pages 719–726, Haifa, Israel, June 2010. Omnipress.

[18] J. Zico Kolter and Andrew Y. Ng. Regularization and feature selection in least-squares temporal difference learning. In *ICML '09: Proceedings of the 26th Annual International Conference on Machine Learning*, pages 521–528. ACM, 2009.

[19] Martin Riedmiller. Neural fitted Q iteration – first experiences with a data efficient neural reinforcement learning method. In *16th European Conference on Machine Learning*, pages 317–328, 2005.

[20] Damien Ernst, Pierre Geurts, and Louis Wehenkel. Tree-based batch mode reinforcement learning. *Journal of Machine Learning Research*, 6:503–556, 2005.

[21] Daniela Pucci de Farias and Benjamin Van Roy. The linear programming approach to approximate dynamic programming. *Operations Research*, 51(6):850–865, 2003.

[22] Marek Petrik and Shlomo Zilberstein. Constraint relaxation in approximate linear programs. In *Proceedings of the 26th Annual International Conference on Machine Learning*, ICML '09, pages 809–816, New York, NY, USA, 2009. ACM.

[23] Rémi Munos. Error bounds for approximate policy iteration. In *ICML 2003: Proceedings of the 20th Annual International Conference on Machine Learning*, pages 560–567, 2003.

[24] Rémi Munos. Performance bounds in $L_p$ norm for approximate value iteration. *SIAM Journal on Control and Optimization*, pages 541–561, 2007.

[25] Amir-massoud Farahmand, Rémi Munos, and Csaba Szepesvári. Error propagation for approximate policy and value iteration. In J. Lafferty, C. K. I. Williams, J. Shawe-Taylor, R.S. Zemel, and A. Culotta, editors, *Advances in Neural Information Processing Systems (NIPS - 23)*, pages 568–576. 2010.

